# Automatic Feature Induction
# for Stagewise Collaborative Filtering

**Joonseok Lee**[a]**, Mingxuan Sun**[a]**, Seungyeon Kim**[a]**, Guy Lebanon**[a, b]
[a] College of Computing, Georgia Institute of Technology, Atlanta, GA 30332
[b] Google Research, Mountain View, CA 94043
{jlee716, msun3, seungyeon.kim}@gatech.edu, lebanon@cc.gatech.edu

## Abstract

Recent approaches to collaborative filtering have concentrated on estimating an algebraic or statistical model, and using the model for predicting missing ratings. In this paper we observe that different models have relative advantages in different regions of the input space. This motivates our approach of using stagewise linear combinations of collaborative filtering algorithms, with non-constant combination coefficients based on kernel smoothing. The resulting stagewise model is computationally scalable and outperforms a wide selection of state-of-the-art collaborative filtering algorithms.

## 1 Introduction

Recent approaches to collaborative filtering (CF) have concentrated on estimating an algebraic or statistical model, and using the model for predicting the missing rating of user $u$ on item $i$. We denote CF methods as $f(u, i)$, and the family of potential CF methods as $\mathcal{F}$.

Ensemble methods, which combine multiple models from $\mathcal{F}$ into a "meta-model", have been a significant research direction in classification and regression. Linear combinations of $K$ models

$$F^{(K)}(x) = \sum_{k=1}^{K} \alpha_k f_k(x) \tag{1}$$

where $\alpha_1, \dots \alpha_K \in \mathbb{R}$ and $f_1, \dots, f_K \in \mathcal{F}$, such as boosting or stagewise linear regression and stagewise logistic regression, enjoy a significant performance boost over the single top-performing model. This is not surprising since (1) includes as a degenerate case each of the models $f \in \mathcal{F}$ by itself. Stagewise models are greedy incremental models of the form

$$(\alpha_k, f_k) = \underset{\alpha_k \in \mathbb{R}, f_k \in \mathcal{F}}{\arg \min} \operatorname{Risk}(F^{(k-1)} + \alpha_k f_k), \qquad k = 1, \dots, K, \tag{2}$$

where the parameters of $F^{(K)}$ are estimated one by one without modifying previously selected parameters). Stagewise models have two important benefits: (a) a significant resistance to overfitting, and (b) computational scalability to large data and high $K$.

It is somewhat surprising that ensemble methods have had relatively little success in the collaborative filtering literature. Generally speaking, ensemble or combination methods have shown only a minor improvement over the top-performing CF methods. The cases where ensemble methods did show an improvement (for example the Netflix prize winner [10] and runner up), relied heavily on manual feature engineering, manual parameter setting, and other tinkering.

This paper follows up on an experimental discovery: different recommendation systems perform better than others for some users and items but not for others. In other words, the relative strengths of two distinct CF models $f_1(u, i), f_2(u, i) \in \mathcal{F}$ depend on the user $u$ and the item $i$ whose rating

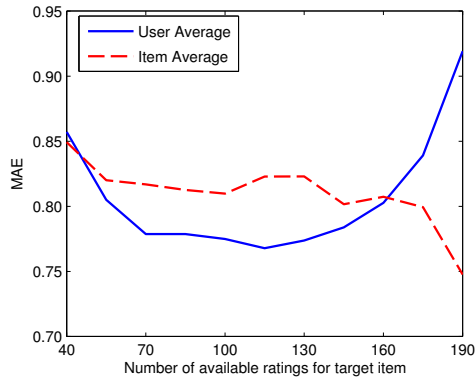

Figure 1: Test set loss (mean absolute error) of two simple algorithms (user average and item average) on items with different number of ratings.

is being predicted. One example of two such systems appears in Figure 1 that graphs the test-set loss of two recommendation rules (user average and item average) as a function of the number of available ratings for the recommended item $i$. The two recommendation rules outperform each other, depending on whether the item in question has few or many ratings in the training data. We conclude from this graph and other comprehensive experiments [14] that algorithms that are inferior in some circumstances may be superior in other circumstances.

The inescapable conclusion is that the weights $\alpha_k$ in the combination should be functions of $u$ and $i$ rather than constants

$$F^{(K)}(u,i) = \sum_{k=1}^{K} \alpha_k(u,i) f_k(u,i) \qquad (3)$$

where $\alpha_k(u,i) \in \mathbb{R}$ and $f_k \in \mathcal{F}$ for $k = 1, \dots, K$. In this paper we explore the use of such models for collaborative filtering, where the weight functions $\alpha_k(u,i)$ are learned from. A major part of our contribution is a feature induction strategy to identify feature functions expressing useful locality information. Our experimental study shows that the proposed method outperforms a wide variety of state-of-the-art and traditional methods, and also outperforms other CF ensemble methods.

## 2   Related Work

Many memory-based CF methods predict the rating of items based on the similarity of the test user and the training users [21, 3, 6]. Similarity measures include Pearson correlation [21] and Vector cosine similarity [3, 6]. Other memory-based CF methods includes item-based CF [25] and a non-parametric probabilistic model based on ranking preference similarities [28].

Model-based CF includes user and item clustering [3, 29, 32], Bayesian networks [3], dependence network [5] and probabilistic latent variable models [19, 17, 33]. Slope-one [16] achieved fast and reasonably accurate prediction. The state-of-the-art methods including the Netflix competition winner are based on matrix factorization. The factorized matrix can be used to fill out the unobserved entries of the user-rating matrix in a way similar to latent factor analysis [20, 12, 9, 13, 24, 23, 11].

Some recent work suggested that combining different CF models may improve the prediction accuracy. Specifically, a memory-based method linearly combined with a latent factor method [1, 8] retains the advantages of both models. Ensembles of maximum margin matrix factorizations were explored to improve the result of a single MMMF model in [4]. A mixture of experts model is proposed in [27] to linearly combine the prediction results of more than two models. In many cases, there is significant manual intervention such as setting the combination weights manually.

Feature-weighted linear stacking [26] is the ensemble method most closely related to our approach. The primary difference is the manual selection of features in [26] as opposed to automatic induction of local features in our paper that leads to a significant improvement in prediction quality. Model combination based on locality has been proposed in other machine learning topics, such as classification [31, 18] or sensitivity estimation [2].

# 3 Combination of CF Methods with Non-Constant Weights

Recalling the linear combination (3) from Section 1, we define non-constant combination weights $\alpha_k(u, i)$ that are functions of the user and item that are being predicted. We propose the following algebraic form

$$\alpha_k(u, i) = \beta_k \, h_k(u, i), \quad \beta_k \in \mathbb{R}, \quad h_k \in \mathcal{H} \tag{4}$$

where $\beta_k$ is a parameter and $h_k$ is a function selected from a family $\mathcal{H}$ of candidate feature functions.

The combination (3) with non-constant weights (4) enables some CF methods $f_k$ to be emphasized for some user-item combinations through an appropriate selection of the $\beta_k$ parameters. We assume that $\mathcal{H}$ contains the constant function, capturing the constant-weight combination within our model.

Substituting (4) into (3) we get

$$F^{(K)}(u, i) = \sum_{k=1}^{K} \beta_k \, h_k(u, i) \, f_k(u, i), \quad \beta_k \in \mathbb{R}, \quad h_k \in \mathcal{H}, \quad f_k \in \mathcal{F}. \tag{5}$$

Note that since $h_k$ and $f_k$ are selected from the sets of CF methods and feature functions respectively, we may have $f_j = f_l$ or $h_j = h_l$ for $j \neq l$. This is similar to boosting and other stagewise algorithms where one feature or base learner may be chosen multiple times, effectively updating its associate feature functions and parameters. The total weight function associated with a particular $f \in \mathcal{F}$ is $\sum_{k:f_k=f} \beta_k h_k(u, i)$.

A simple way to fit $\beta = (\beta_1, \ldots, \beta_K)$ is least squares

$$\beta^* = \arg\min_{\beta \in C} \sum_{u,i} \left( F^{(K)}(u, i) - R_{u,i} \right)^2, \tag{6}$$

where $R_{u,i}$ denotes the rating of user $u$ on item $i$ in the training data and the summation ranges over all ratings in the training set. A variation of (6), where $\beta$ is constrained such that $\alpha_k(u, i) \geq 0$, and $\sum_{k=1}^{K} \alpha_k(u, i) = 1$ endows $F$ with the following probabilistic interpretation

$$F(u, i) = \mathsf{E}_p \left\{ f \mid u, i \right\}, \tag{7}$$

where $f$ represents a random draw from $\mathcal{F}$, with probabilities $p(f|u, i)$ proportional to $\sum_{k:f_k=f} \beta_k h_k(u, i)$. In contrast to standard combination models with fixed weights, (7) forms a conditional expectation, rather than an expectation.

# 4 Inducing Local Features

In contrast to [26] that manually defined 25 features, we induce the features $h_k$ from data. The features $h_k(u, i)$ should emphasize users $u$ and items $i$ that are likely to lead to variations in the relative strength of the $f_1, \ldots, f_K$. We consider below two issues: (i) defining the set $\mathcal{H}$ of candidate features, and (ii) a strategy for selecting features from $\mathcal{H}$ to add to the combination $F$.

## 4.1 Candidate Feature Families $\mathcal{H}$

We denote the sets of users and items by $U$ and $I$ respectively, and the domain of $f \in \mathcal{F}$ and $h \in \mathcal{H}$ as $\Omega = U \times I$. The set $R \subset \Omega$ is the set of user-item pairs present in the training set, and the set of user-item pairs that are being predicted is $\omega \in \Omega \setminus R$.

We consider the following three unimodal functions on $\Omega$, parameterized by a location parameter or mode $\omega^* = (u^*, i^*) \in \Omega$ and a bandwidth $h > 0$

$$K_{h,(u^*,i^*)}^{(1)}(u, i) \propto \left( 1 - \frac{d(u^*, u)}{h} \right) I\left( d(u^*, u) \leq h \right),$$

$$K_{h,(u^*,i^*)}^{(2)}(u, i) \propto \left( 1 - \frac{d(i^*, i)}{h} \right) I\left( d(i^*, i) \leq h \right),$$

$$K_{h,(u^*,i^*)}^{(3)}(u, i) \propto \left( 1 - \frac{d(u^*, u)}{h} \right) I\left( d(u^*, u) \leq h \right) \cdot \left( 1 - \frac{d(i^*, i)}{h} \right) I\left( d(i^*, i) \leq h \right), \tag{8}$$

where $I(A) = 1$ if $A$ holds and 0 otherwise. The first function is unimodal in $u$, centered around $u^*$, and constant in $i$. The second function is unimodal in $i$, centered around $i^*$, and constant in $u$. The third is unimodal in $u, i$ and centered around $(u^*, i^*)$.

There are several possible choices for the distance functions in (8) between users and between items. For simplicity, we use in our experiments the angular distance

$$d(x, y) = \arccos \left( \frac{\langle x, y \rangle}{\|x\| \cdot \|y\|} \right) \tag{9}$$

where the inner products above are computed based on the user-item rating matrix expressing the training set (ignoring entries not present in both arguments).

The functions (8) are the discrete analogs of the triangular kernel $K_h(x) = h^{-1}(1 - |x - x^*|/h)I(|x - x^*| \leq h)$ used in non-parametric kernel smoothing [30]. Their values decay linearly with the distance from their mode (truncated at zero), and feature a bandwidth parameter $h$, controlling the rate of decay. As $h$ increases the support size $|\{\omega \in \Omega : K(\omega) > 0\}|$ increases and $\max_{\omega \in \Omega} K(\omega)$ decreases.

The unimodal feature functions (8) capture locality in the $\Omega$ space by measuring proximity to a mode, representing a user $u^*$, an item $i^*$, or a user-item pair. We define the family of candidate features $\mathcal{H}$ as all possible additive mixtures or max-mixtures of the functions (8), parameterized by a set of multiple modes $\boldsymbol{\omega}^* = \{\omega_1^*, \ldots, \omega_r^*\}$

$$K_{\boldsymbol{\omega}^*}(u, i) \propto \sum_{j=1}^{r} K_{\omega_j^*}(u, i) \tag{10}$$

$$K_{\boldsymbol{\omega}^*}(u, i) \propto \max_{j=1,\ldots,r} K_{\omega_j^*}(u, i). \tag{11}$$

Using this definition, features functions $h_k(u, i) \in \mathcal{H}$ are able to express a wide variety of locality information involving multiple potential modes.

We discuss next the strategy for identifying useful features from $\mathcal{H}$ and adding them to the model $F$ in a stagewise manner.

## 4.2 Feature Induction Strategy

Adapting the stagewise learning approach to the model (5) we have

$$F^{(K)}(u, i) = \sum_{k=1}^{K} \beta_k \, h_k(u, i) \, f_k(u, i), \tag{12}$$

$$(\beta_k, h_k, f_k) = \underset{\beta_k \in \mathbb{R}, h_k \in \mathcal{H}, f_k \in \mathcal{F}}{\arg\min} \sum_{(u,i) \in R} \left( F^{(k-1)}(u, i) + \beta_k h_k(u, i) f_k(u, i) - R_{u,i} \right)^2.$$

It is a well-known fact that stagewise algorithms sometimes outperform non-greedy algorithms due to resistance to overfitting (see [22], for example). This explains the good generalization ability of boosting and stage-wise linear regression.

From a computational standpoint, (12) scales nicely with $K$ and with the training set size. The one-dimensional quadratic optimization with respect to $\beta$ is solved via a closed form, but the optimization over $\mathcal{F}$ and $\mathcal{H}$ has to be done by brute force or by some approximate method such as sampling. The computational complexity of each iteration is thus $O(|\mathcal{H}| \cdot |\mathcal{F}| \cdot |R|)$, assuming no approximation are performed.

Since we consider relatively small families $\mathcal{F}$ of CF methods, the optimization over $\mathcal{F}$ does not pose a substantial problem. The optimization over $\mathcal{H}$ is more problematic since $\mathcal{H}$ is potentially infinite, or otherwise very large. We address this difficulty by restricting $\mathcal{H}$ to a finite collection of additive or max-mixtures kernels with $r$ modes, randomly sampled from the users or items present in the training data. Our experiments conclude that it is possible to find useful features from a surprisingly small number of randomly-chosen samples.

## 5 Experiments

We describe below the experimental setting, followed by the experimental results and conclusions.

### 5.1 Experimental Design

We used a recommendation algorithm toolkit PREA [15] for candidate algorithms, including three simple baselines (Constant model, User Average, and Item Average) and five matrix-factorization methods (Regularized SVD, NMF [13], PMF [24], Bayesian PMF [23], and Non-Linear PMF [12]), and Slope-one [16]. This list includes traditional baselines as well as state-of-the-art CF methods that were proposed recently in the research literature. We evaluate the performance using the Root Mean Squared Error (RMSE), measured on the test set.

Table 1 lists 5 experimental settings. SINGLE runs each CF algorithm individually, and chooses the one with the best average performance. CONST combines all candidate algorithms with constant weights as in (1). FWLS combines all candidate algorithms with non-constant weights as in (3) [26]. For CONST and FWLS, the weights are estimated from data by solving a least-square problem. STAGE combines CF algorithms in stage-wise manner. FEAT applies the feature induction techniques discussed in Section 4.

To evaluate whether the automatic feature induction in FEAT works better or worse than manually constructed features, we used in FWLS and STAGE manual features similar to the ones in [26] (excluding features requiring temporal data). Examples include number of movies rated per user, number of users rating each movie, standard deviation of the users' ratings, and standard deviation of the item's ratings.

The feature induction in FEAT used a feature space $\mathcal{H}$ with additive multi-mode smoothing kernels as described in Section 4 (for simplicity we avoided kernels unimodal in both $u$ and $i$). The family $\mathcal{H}$ included 200 randomly sampled features (a new sample was taken for each of the iterations in the stagewise algorithms). The $r$ in (11) was set to 5% of user or item count, and bandwidth $h$ values of 0.05 (an extreme case where most features have value either 0 or 1) and 0.8 (each user or item has moderate similarity values). The stagewise algorithm continues until either five consecutive trials fail to improve the RMSE on validation set, or the iteration number reaches 100, which occur only in a few cases. We used similar $L_2$ regularization for all methods (both stagewise and non-stagewise), where the regularization parameter was selected among 5 different values based on a validation set.

We experimented with the two standard MovieLens datasets: 100K and 1M, and with the Netflix dataset. In the Netflix dataset experiments, we sub-sampled the data since (a) running state-of-the-art candidate algorithms on the full Netflix data takes too long time - for example, Bayesian PMF was reported to take 188 hours [23], and (b) it enables us to run extensive experiments measuring the performance of the CF algorithms as a function of the number of users, number of items, voting sparsity, and facilitates cross-validation and statistical tests. More specifically, we sub-sampled from the most active $M$ users and the most often rated $N$ items to obtain pre-specified data density levels $|R|/|\Omega|$. As shown in Table 2, we varied either the user or item count in the set $\{1000, 1500, 2000, 2500, 3000\}$, holding the other variable fixed at 1000 and the density at 1%, which is comparable density of the original Netflix dataset. We also conducted an experiment where the data density varied in the set $\{1\%, 1.5\%, 2\%, 2.5\%\}$ with fixed user and item count of 1000 each.

We set aside a randomly chosen 20% for test set, and used the remaining 80% for both for training the individual recommenders and for learning the ensemble model. It is possible, and perhaps more motivated, to use two distinct train sets for the CF models and the ensemble. However, in our case, we got high performance even in the case of using the same training dataset in both stages.

| Method | C | W | S | I | Explanation |
|--------|---|---|---|---|-------------|
| SINGLE |   |   |   |   | Best-performed single CF algorithm |
| CONST  | O |   |   |   | Mixture of CF without features |
| FWLS   | O | O |   |   | Mixture of CF with manually-designed features |
| STAGE  | O | O | O |   | Stagewise mixture with manual features |
| FEAT   | O | O | O | O | Stagewise mixture with induced features |

Table 1: Experimental setting. (C: Combination of multiple algorithms, W: Weights varying with features, S: Stage-wise algorithm, I: Induced features)

| Dataset | | Netflix | | | | | | | | MovieLens | |
|---|---|---|---|---|---|---|---|---|---|---|---|
| User Count | | 1000 | 2000 | 3000 | 1000 | | 1000 | | | 943 | 6039 |
| Item Count | | 1000 | 1000 | | 2000 | 3000 | 1000 | | | 1682 | 3883 |
| Density | | 1.0% | 1.0% | | 1.0% | | 1.5% | 2.0% | 2.5% | 6.3% | 4.3% |
| Single CF | Constant | 1.2188 | 1.2013 | 1.2072 | 1.1964 | 1.1888 | 1.2188 | 1.2235 | 1.2113 | 1.2408 | 1.2590 |
| | UserAvg | 1.0566 | 1.0513 | 1.0375 | 1.0359 | *1.0174* | 1.0566 | 1.0318 | 1.0252 | 1.0408 | 1.0352 |
| | ItemAvg | 1.1260 | 1.0611 | 1.0445 | 1.1221 | 1.1444 | 1.1260 | 1.1029 | 1.0900 | 1.0183 | 0.9789 |
| | Slope1 | 1.4490 | 1.4012 | 1.3321 | 1.4049 | 1.3196 | 1.4490 | 1.3505 | 1.0725 | 0.9371 | 0.9017 |
| | RegSVD | 1.0623 | *1.0155* | 1.0083 | *1.0354* | 1.0289 | *1.0343* | 1.0154 | *1.0020* | *0.9098* | *0.8671* |
| | NMF | 1.0784 | 1.0205 | *1.0069* | 1.0423 | 1.0298 | 1.0406 | *1.0151* | 1.0091 | 0.9601 | 0.9268 |
| | PMF | 1.6180 | 1.4824 | 1.4081 | 1.4953 | 1.4804 | 1.4903 | 1.3594 | 1.1818 | 0.9328 | 0.9623 |
| | BPMF | 1.3973 | 1.2951 | 1.2949 | 1.2566 | 1.2102 | 1.3160 | 1.2021 | 1.1514 | 0.9629 | 0.9000 |
| | NLPMF | *1.0561* | 1.0507 | 1.0382 | 1.0361 | 1.0471 | 1.0436 | 1.0382 | 1.0523 | 0.9560 | 0.9415 |
| Combined | SINGLE | 1.0561 | 1.0155 | 1.0069 | 1.0354 | 1.0174 | 1.0343 | 1.0151 | 1.0020 | 0.9098 | 0.8671 |
| | CONST | 1.0429 | 1.0072 | 0.9963 | 1.0198 | 1.0255 | 0.9968 | 0.9824 | | 0.9073 | 0.8660 |
| | FWLS | 1.0288 | 1.0050 | 0.9946 | 1.0089 | 1.0016 | 1.0179 | 0.9935 | 0.9802 | 0.9010 | 0.8649 |
| | STAGE | 1.0036 | 0.9784 | 0.9668 | 0.9967 | 0.9821 | 0.9935 | 0.9846 | 0.9769 | 0.8961 | 0.8623 |
| | FEAT | *0.9862* | *0.9607* | *0.9607* | *0.9740* | *0.9717* | *0.9703* | *0.9589* | *0.9492* | *0.8949* | *0.8569* |
| p-Value | | 0.0028 | 0.0001 | 0.0003 | 0.0008 | 0.0014 | 0.0002 | 0.0019 | 0.0013 | 0.0014 | 0.0023 |

Table 2: Test error in RMSE (lower values are better) for single CF algorithms used as candidates and combined models. Data where $M$ or $N$ is 1500 or 2500 are omitted due to the lack of space, as it is shown in Figure 2. The best-performing one in each group is indicated in *Italic*. The last row indicates $p$-value for statistical test of hypothesis FEAT $\succ$ FWLS.

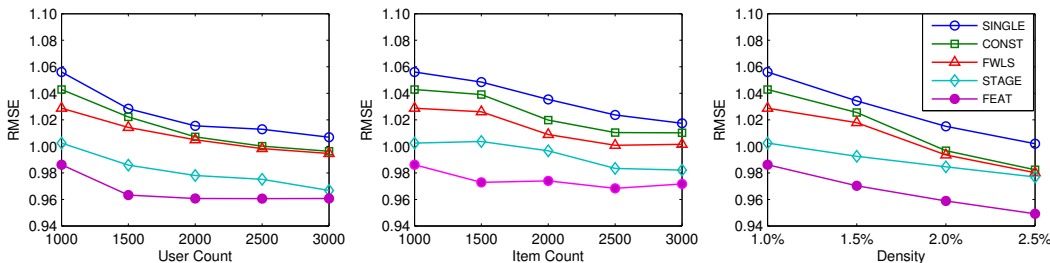

Figure 2: Performance trend with varied user count (left), item count (middle), and density (right) on Netflix dataset.

For stagewise methods, the 80% train set was divided to 60% training set and 20% validation set, used to determine when to stop the stagewise addition process. The non-stagewise methods used the entire 80% for training. The 10% of training set is used to select regularization parameter for both stagewise and non-stagewise. The results were averaged over 10 random data samples.

## 6   Result and Discussion

### 6.1   Performance Analysis and Example

Table 2 displays the performance in RMSE of each combination method, as well as the individual algorithms. Examining it, we observe the following partial order with respect to prediction accuracy: FEAT $\succ$ STAGE $\succ$ FWLS $\succ$ CONST $\succ$ SINGLE.

- **FWLS $\succ$ CONST $\succ$ SINGLE**: Combining CF algorithms (even only with constant weights) produces better prediction than the best-single CF method. Also, using non-constant weights improves performance further. This result is consistent with what has been known in literature [7, 26].

- **STAGE $\succ$ FWLS**: Figure 2 indicates that stagewise combinations where features are chosen with replacement are more accurate. The selection with replacement allow certain features to be selected more than once, correcting a previous inaccurate parameter setting.

- **FEAT $\succ$ STAGE**: Making use of induced features improves prediction accuracy further from stagewise optimization with manually-designed features.

Overall, our experiments indicate that the combination with non-constant weights and feature induction (FEAT) outperforms three baselines (the best single method, standard combinations with constant weight, and the FWLS method using manually constructed features [26]). We tested the

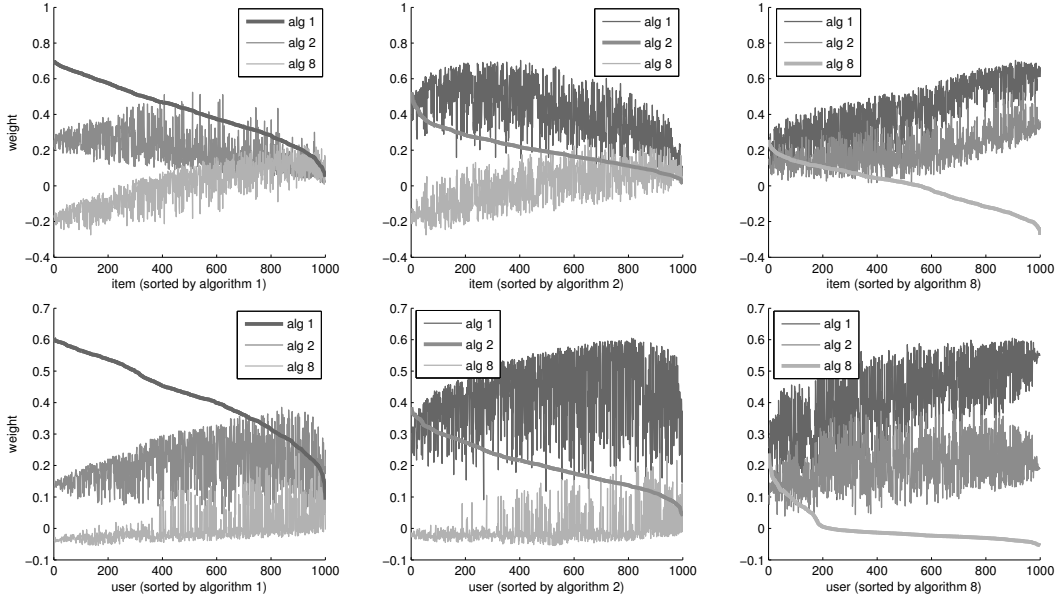

Figure 3: Average weight values of each item (top) and user (bottom) sorted in the order of high to low weight of selected algorithm. Note that the sorting order is similar between algorithm 1 (User Average) and 2 (Item Average). In contrast, algorithm 8 (NLPMF) has opposite order, which will be weighted higher in different part of the data, compared to algorithm 1 and 2.

hypothesis $RMSE_{FEAT} < RMSE_{FWLS}$ with paired $t$-test. Based on the $p$-values (See the last row in Table 2), we can reject the null hypothesis with significance of 99%. We conclude that our proposed combination outperforms state-of-the-art methods, and several previously proposed combination methods.

To see how feature induction works in detail, we illustrate an example with a case where the user count and item count equals 1000. Figure 3 shows the average weight distribution that each user or item receives under three CF methods: user average, item average, and NLPMF. We focused on these three methods since they are frequently selected by the stagewise algorithm. The $x$ axis variables in the three panels are sorted in the order of decreasing weights of selected algorithm. Note that in each figure, one curve is monotonically decaying, showing the weights of the CF method according to which the sorting was done. An interesting observation is that algorithm 1 (User Average) and algorithm 2 (Item Average) have similar pattern of sorting order in Figure 3 (right column). In other words, these two algorithms are similar in nature, and are relatively strong or weaker in similar regions of $\Omega$. Algorithm 8 (NLPMF) on the other hand, has a very different relative strength pattern.

## 6.2 Trend Analysis

Figure 2 graphs the RMSE of the different combination methods as a function of the user count, item count, and density. We make the following observations.

- As expected, prediction accuracy for all combination methods and for the top single method improves with the user count, item count, and density.

- The performance gap between the best single algorithm and the combinations tends to decrease with larger user and item count. This is a manifestation of the law of diminishing returns, and the fact that the size of a suitable family $\mathcal{H}$ capturing locality information increases with the user and item count. Thus, the stagewise procedure becomes more challenging computationally, and less accurate since in our experiment we sampled same number of compositions from $\mathcal{H}$, rather than increasing it for larger data.

- We note that all combination methods and the single best CF method improve performance as the density increases. The improvement seems to be the most pronounced for the single best algorithm and for the FEAT method, indicating that FEAT scales up its performance aggressively with increasing density levels.

- Comparing the left and middle panels of Figure 2 implies that having more users is more informative than having more items. In other words, if the total dataset size $M \times N$ is equal, the performance tends to be better when $M > N$ (left panel of Figure 2) than $M < N$ (middle panel of Figure 2).

## 6.3 Scalability

Our proposed stagewise algorithm is very efficient, when compared to other feature selection algorithms such as step-wise or subset selection. Nevertheless, the large number of possible features may result in computational issues. In our experiments, we sampled from the space of candidate features a small subset of features that was considered for addition (the random subset is different in each iteration of the stagewise algorithm). In the limit $K \rightarrow \infty$, such a sampling scheme would recover the optimal ensemble as each feature will be selected for consideration infinitely often. Our experiments conclude that this scheme works well also in practice and results in significant improvement to the state-of-the-art even for a relatively small sample of feature candidates such as 200. Viewed from another perspective, this implies that randomly selecting such a small subset of features each iteration ensures the selection of useful features. In fact, the features induced in this manner were found to be more useful than the manually crafted features in the FWLS algorithm [26].

## 7 Summary

We started from an observation that the relative performance of different candidate recommendation systems $f(u, i)$ depends on $u$ and $i$, for example on the activity level of user $u$ and popularity of item $i$. This motivated the development of combination of recommendation systems with non-constant weights that emphasize different candidates based on their relative strengths in the feature space. In contrast to the FWLS method that focused on manual construction of features, we developed a feature induction algorithm that works in conjunction with stagewise least-squares. We formulate a family of feature function, based on the discrete analog of triangular kernel smoothing. This family captures a wide variety of local information and is thus able to model the relative strengths of the different CF methods and how they change across $\Omega$.

The combination with induced features outperformed any of the base candidates as well as other combination methods in literature. This includes the recently proposed FWLS method that uses manually constructed feature function. As our candidates included many of the recently proposed state-of-the-art recommendation systems our conclusions are significant for the engineering community as well as recommendation system scientists.

## References

[1] R. Bell, Y. Koren, and C. Volinsky. Modeling relationships at multiple scales to improve accuracy of large recommender systems. In *Proc. of the ACM SIGKDD*, 2007.

[2] P. Bennett. Neighborhood-based local sensitivity. In *Proc. of the European Conference on Machine Learning*, 2007.

[3] J. Breese, D. Heckerman, and C. Kadie. Empirical analysis of predictive algorithms for collaborative filtering. In *Uncertainty in Artificial Intelligence*, 1998.

[4] D. DeCoste. Collaborative prediction using ensembles of maximum margin matrix factorizations. In *Proc. of the International Conference on Machine Learning*, 2006.

[5] D. Heckerman, D. Maxwell Chickering, C. Meek, R. Rounthwaite, and C. Kadie. Dependency networks for inference, collaborative filtering, and data visualization. *Journal of Machine Learning Research*, 1, 2000.

[6] J. L. Herlocker, J. A. Konstan, A. Borchers, and J. Riedl. An algorithmic framework for performing collaborative filtering. In *Proc. of ACM SIGIR Conference*, 1999.

[7] M. Jahrer, A. Töscher, and R. Legenstein. Combining predictions for accurate recommender systems. In *Proc. of the ACM SIGKDD*, 2010.

[8] Y. Koren. Factorization meets the neighborhood: a multifaceted collaborative filtering model. In *Proc. of the ACM SIGKDD*, 2008.

[9] Y. Koren. Factor in the neighbors: Scalable and accurate collaborative filtering. *ACM Transactions on Knowledge Discovery from Data*, 4(1):1–24, 2010.

[10] Y. Koren, R. Bell, and C. Volinsky. Matrix factorization techniques for recommender systems. *Computer*, 42(8):30–37, 2009.

[11] B. Lakshminarayanan, G. Bouchard, and C. Archambeau. Robust bayesian matrix factorisation. In *Proc. of the International Conference on Artificial Intelligence and Statistics*, 2011.

[12] N. D. Lawrence and R. Urtasun. Non-linear matrix factorization with gaussian processes. In *Proc. of the International Conference on Machine Learning*, 2009.

[13] D. Lee and H. Seung. Algorithms for non-negative matrix factorization. In *Advances in Neural Information Processing Systems*, 2001.

[14] J. Lee, M. Sun, and G. Lebanon. A comparative study of collaborative filtering algorithms. *ArXiv Report 1205.3193*, 2012.

[15] J. Lee, M. Sun, and G. Lebanon. Prea: Personalized recommendation algorithms toolkit. *Journal of Machine Learning Research*, 13:2699–2703, 2012.

[16] D. Lemire and A. Maclachlan. Slope one predictors for online rating-based collaborative filtering. *Society for Industrial Mathematics*, 5:471–480, 2005.

[17] B. Marlin. Modeling user rating profiles for collaborative filtering. In *Advances in Neural Information Processing Systems*, 2004.

[18] C. J. Merz. Dynamical selection of learning algorithms. *Lecture Notes in Statistics*, pages 281–290, 1996.

[19] D. M. Pennock, E. Horvitz, S. Lawrence, and C. L. Giles. Collaborative filtering by personality diagnosis: A hybrid memory- and model-based approach. In *Uncertainty in Artificial Intelligence*, 2000.

[20] J.D.M. Rennie and N. Srebro. Fast maximum margin matrix factorization for collaborative prediction. In *Proc. of the International Conference on Machine Learning*, 2005.

[21] P. Resnick, N. Iacovou, M. Suchak, P. Bergstrom, and J. Riedl. Grouplens: an open architecture for collaborative filtering of netnews. In *Proc. of the Conference on CSCW*, 1994.

[22] L. Reyzin and R. E. Schapire. How boosting the margin can also boost classifier complexity. In *Proc. of the International Conference on Machine Learning*, 2006.

[23] R. Salakhutdinov and A. Mnih. Bayesian probabilistic matrix factorization using markov chain monte carlo. In *Proc. of the International Conference on Machine Learning*, 2008.

[24] R. Salakhutdinov and A. Mnih. Probabilistic matrix factorization. In *Advances in Neural Information Processing Systems*, 2008.

[25] B. Sarwar, G. Karypis, J. Konstan, and J. Reidl. Item-based collaborative filtering recommendation algorithms. In *Proc. of the International Conference on World Wide Web*, 2001.

[26] J. Sill, G. Takacs, L. Mackey, and D. Lin. Feature-weighted linear stacking. *Arxiv Report arXiv:0911.0460*, 2009.

[27] X. Su, R. Greiner, T. M. Khoshgoftaar, and X. Zhu. Hybrid collaborative filtering algorithms using a mixture of experts. In *Proc. of the IEEE/WIC/ACM International Conference on Web Intelligence*, 2007.

[28] M. Sun, G. Lebanon, and P. Kidwell. Estimating probabilities in recommendation systems. In *Proc. of the International Conference on Artificial Intelligence and Statistics*, 2011.

[29] L. H. Ungar and D. P. Foster. Clustering methods for collaborative filtering. In *AAAI Workshop on Recommendation Systems*, 1998.

[30] M. P. Wand and M. C. Jones. *Kernel Smoothing*. Chapman and Hall/CRC, 1995.

[31] K. Woods, W.P. Kegelmeyer Jr, and K. Bowyer. Combination of multiple classifiers using local accuracy estimates. *IEEE Transactions on Pattern Analysis and Machine Intelligence*, 19(4):405–410, 1997.

[32] G. R. Xue, C. Lin, Q. Yang, W. S. Xi, H. J. Zeng, Y. Yu, and Z. Chen. Scalable collaborative filtering using cluster-based smoothing. In *Proc. of ACM SIGIR Conference*, 2005.

[33] K. Yu, S. Zhu, J. Lafferty, and Y. Gong. Fast nonparametric matrix factorization for large-scale collaborative filtering. In *Proc. of ACM SIGIR Conference*, 2009.

